# Different Cortico-Basal Ganglia Loops Specialize in Reward Prediction on Different Time Scales

**Saori Tanaka**            **Kenji Doya**
Nara Institute of Science and Technology
ATR Computational Neuroscience Laboratories
CREST, Japan Science and Technology Corporation
Kyoto, Japan
*xsaori@atr.co.jp*            *doya@atr.co.jp*

**Go Okada**     **Kazutaka Ueda**     **Yasumasa Okamoto**     **Shigeto Yamawaki**
Hiroshima University School of Medicine
CREST, Japan Science and Technology Corporation
Hiroshima, Japan

## Abstract

To understand the brain mechanisms involved in reward prediction on different time scales, we developed a *Markov decision task* that requires prediction of both immediate and future rewards, and analyzed subjects' brain activities using functional MRI. We estimated the time course of reward prediction and reward prediction error on different time scales from subjects' performance data, and used them as the explanatory variables for SPM analysis. We found topographic maps of different time scales in medial frontal cortex and striatum. The result suggests that different cortico-basal ganglia loops are specialized for reward prediction on different time scales.

## 1   Introduction

In our daily life, we make decisions based on the prediction of rewards on different time scales; immediate and long-term effects of an action are often in conflict, and biased evaluation of immediate or future outcome can lead to pathetic behaviors.

Lesions in the central serotonergic system result in impulsive behaviors in humans [1], and animals [2, 3], which  can be attributed to deficits in reward prediction on a long time scale. Damages in the ventral part of medial frontal cortex (MFC) also cause deficits in decision-making that requires assessment of future outcomes [4-6].

A possible mechanism underlying these observations is that different brain areas are specialized for reward prediction on different time scales, and that the ascending

serotonergic system activates those specialized for predictions in longer time scales [7].

The theoretical framework of temporal difference (TD) learning [8] successfully explains reward-predictive activities of the midbrain dopaminergic system as well as those of the cortex and the striatum [9-13]. In TD learning theory, the predicted amount of future reward starting from a state $s(t)$ is formulated as the "value function"

$$V(t) = E[r(t + 1) + \gamma r(t + 2) + \gamma^2 r(t + 3) + \ldots] \qquad (1)$$

and learning is based on the TD error

$$\delta(t) = r(t) + \gamma V(t) - V(t - 1). \qquad (2)$$

The 'discount factor' $\gamma$ controls the time scale of prediction; while only the immediate reward $r(t + 1)$ is considered with $\gamma = 0$, rewards in the longer future are taken into account with $\gamma$ closer to 1.

In order to test the above hypothesis [7], we developed a reinforcement learning task which requires a large value of discount factor for successful performance, and analyzed subjects' brain activities using functional MRI. In addition to conventional block-design analysis, a novel model-based regression analysis revealed topographic representation of prediction time scale with in the cortico-basal ganglia loops.

## 2　Methods

### 2.1　Markov Decision Task

In the Markov decision task (Fig. 1), markers on the corners of a square present four states, and the subject selects one of two actions by pressing a button ($a_1$ = left button, $a_2$ = right button) (Fig. 1A). The action determines both the amount of reward and the movement of the marker (Fig. 1B). In the REGULAR condition, the next trial is started from the marker position at the end of the previous trial. Therefore, in order to maximize the reward acquired in a long run, the subject has to select an action by taking into account both the immediate reward and the future reward expected from the subsequent state. The optimal behavior is to receive small negative rewards at states $s_2$, $s_3$, and $s_4$ to obtain a large positive reward at state $s_1$ (Fig. 1C). In the RANDOM condition, next trial is started from a random marker position so that the subject has to consider only immediate reward. Thus, the optimal behavior is to collect a larger reward at each state (Fig. 1D). In the baseline condition (NO condition), the reward is always zero.

In order to learn the optimal behaviors, the discount factor $\gamma$ has to be larger than 0.3425 in REGULAR condition, while it can be arbitrarily small in RANDOM condition.

### 2.2　fMRI imaging

Eighteen healthy, right-handed volunteers (13 males and 5 females), gave informed consent to take part in the study, with the approval of the ethics and safety committees of ATR and Hiroshima University.

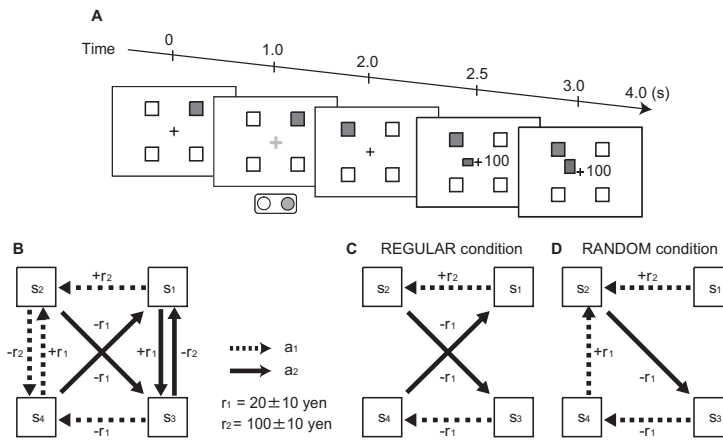

**Fig. 1. (A)** Sequence of stimulus and response events in the Markov decision task. First, one of four squares representing present *state* turns green (0s). As the fixation point turns green (1s), the subject presses either the right or left button within 1 second. After 1s delay, the green square changes its position (2s), and then a reward for the current action is presented by a number (2.5s) and a bar graph showing cumulative reward during the block is updated (3.0s). One trial takes four seconds. Subjects performed five trials in the NO condition, 32 trials in the RANDOM condition, five trials in the NO condition, and 32 trials in the REGULAR condition in one block. They repeated four blocks; thus, the entire experiment consisted of 312 trials, taking about 20 minutes. **(B)** The rule of the reward and marker movement. **(C)** In the REGULAR condition, the optimal behavior is to receive small negative rewards $-r_1$ (-10, -20, or -30 yen) at states $s_2, s_3,$ and $s_4$ to obtain a large positive reward $+r_2$ (90, 100, or 110 yen) at state $s_1$. **(D)** In the RANDOM condition, the next trial is started from random state. Thus, the optimal behavior is to select a larger reward at each state.

A 1.5-Tesla scanner (Marconi, MAGNEX ECLIPSE, Japan) was used to acquire both structural T1-weighted images (TR = 12 s, TE = 450 ms, flip angle = 20 deg, matrix = 256 × 256, FoV = 256 mm, thickness = 1 mm, slice gap = 0 mm ) and T2*-weighted echo planar images (TR = 4 s, TE = 55 msec, flip angle = 90 deg, 38 transverse slices, matrix = 64 × 64, FoV = 192 mm, thickness = 4 mm, slice gap = 0 mm, slice gap = 0 mm) with blood oxygen level-dependent (BOLD) contrast.

## 2.3  Data analysis

The data were preprocessed and analyzed with SPM99 (Friston et al., 1995; Wellcome Department of Cognitive Neurology, London, UK). The first three volumes of images were discarded to avoid T1 equilibrium effects. The images were realigned to the first image as a reference, spatially normalized with respect to the Montreal Neurological Institute EPI template, and spatially smoothed with a Gaussian kernel (8 mm, full-width at half-maximum).

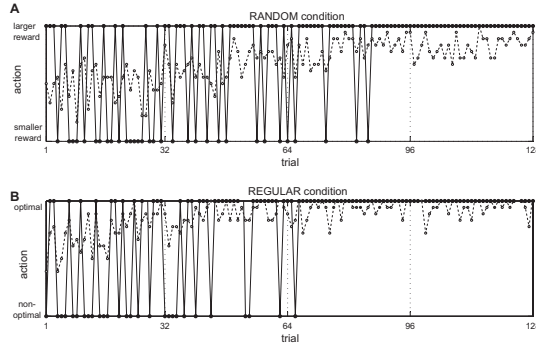

**Fig. 2.** The selected action of a representative single subject (solid line) and the group average ratio of selecting optimal action (dashed line) in **(A)** RANDOM and **(B)** REGULAR conditions.

Images of parameter estimates for the contrast of interest were created for each subject. These were then used for a second-level group analysis using a one-sample t-test across the subjects (random effects analysis).

We conducted two types of analysis. One was block design analysis using three boxcar regressors convolved with a hemodynamic response function as the reference waveform for each condition (RANDOM, REGULAR, and NO). The other was multivariate regression analysis using explanatory variables, representing the time course of the reward prediction $V(t)$ and reward prediction error $\delta(t)$ estimated from subjects' performance data (described below), in addition to three regressors representing the condition of the block.

## 2.4 Estimation of predicted reward $V(t)$ and prediction error $\delta(t)$

The time course of reward prediction $V(t)$ and reward prediction error $\delta(t)$ were estimated from each subject's performance data, i.e. state $s(t)$, action $a(t)$, and reward $r(t)$, as follows.

If the subject starts from a state $s(t)$ and comes back to the same state after $k$ steps, the expected cumulative reward $V(t)$ should satisfy the consistency condition

$$V(t) = r(t + 1) + \gamma\, r(t + 2) + \ldots + \gamma^{k-1} r(t + k) + \gamma^k V(t). \qquad (3)$$

Thus, for each time $t$ of the data file, we calculated the weighted sum of the rewards acquired until the subject returned to the same state and estimated the value function for that episode as

$$\hat{V}(t) = \frac{\left[ r(t+1) + \gamma r(t+2) + \ldots + \gamma^{k-1} r(t+k) \right]}{1 - \gamma^k}. \qquad (4)$$

The estimate of the value function $V(t)$ at time $t$ was given by the average of all previous episodes from the same state as at time $t$

$$V(t) = \frac{1}{L} \sum_{l=1}^{L} \hat{V}(t_l), \qquad (5)$$

where $\{t_1, \ldots, t_L\}$ are the indices of time visiting the same state as $s(t)$, i.e. $s(t_1) = \ldots = s(t_L) = s(t)$. The TD error was given by the difference between the actual reward $r(t)$ and the temporal difference of the value function $V(t)$ according to equation (2).

Assuming that different brain areas are involved in reward prediction on different time scales, we varied the discount factor $\gamma$ as 0, 0.3, 0.6, 0.8, 0.9, and 0.99.

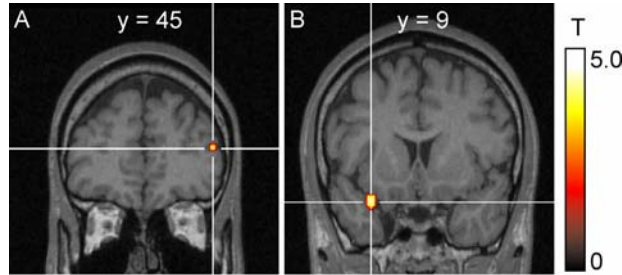

**Fig. 3. (A)** In REGULAR vs. RANDOM comparison, significant activation was observed in DLPFC ((x, y, z) = (46, 45, 9), peak t = 4.06) ($p < 0.001$ uncorrected). **(B)** In RANDOM vs. REGULAR comparison, significant activation was observed in lateral OFC ((x, y, z) = (-32, 9, -21), peak t = 4.90) ($p < 0.001$ uncorrected).

## 3   Results

### 3.1   Behavioral results

Figure 2 summarizes the learning performance of a representative single subject (solid line) and group average (dashed line) during fMRI measurement. Fourteen subjects successfully learned to take larger immediate rewards in the RANDOM condition (Fig. 2A) and a large positive reward at $s_1$ after small negative rewards at $s_2$, $s_3$ and $s_4$ in the REGULAR condition (Fig. 2B).

### 3.2   Block-design analysis

In REGULAR vs. RANDOM contrast, we observed a significant activation in the dorsolateral prefrontal cortex (DLPFC) (Fig. 3A) (p < 0.001 uncorrected). In RANDOM vs. REGULAR contrast, we observed a significant activation in lateral orbitofrontal cortex (lOFC) (Fig. 3B) (p < 0.001 uncorrected).

The result of block-design analysis suggests differential involvement of neural pathways in reward prediction on long and short time scales. The result in RANDOM vs. REGULAR contrast was consistent with previous studies that the OFC is involved in reward prediction within a short delay and reward outcome [14-20].

### 3.3   Regression analysis

We observed significant correlation with reward prediction $V(t)$ in the MFC, DLPFC (all $\gamma$), ventromedial insula (small $\gamma$), dorsal striatum, amygdala, hippocampus, and parahippocampal gyrus (large $\gamma$) (p < 0.001 uncorrected) (Fig. 4A). We also found significant correlation with reward prediction error $\delta(t)$ in the IPC, PMd, cerebellum (all $\gamma$), ventral striatum (small $\gamma$), and lateral OFC (large $\gamma$) (p < 0.001 uncorrected) (Fig. 4B). As we changed the time scale parameter $\gamma$ of reward prediction, we found rostro-caudal maps of correlation to $V(t)$ in MFC with increasing $\gamma$.

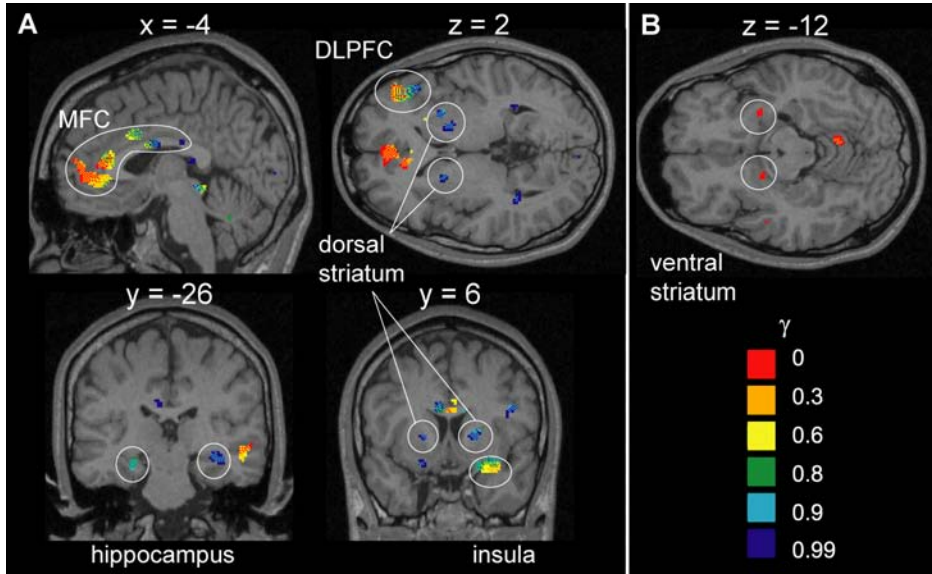

**Fig. 4.** Voxels with a significant correlation ($p < 0.001$ uncorrected) with reward prediction $V(t)$ and prediction error $\delta(t)$ are shown in different colors for different settings of the time scale parameter ($\gamma = 0$ in red, $\gamma = 0.3$ in orange, $\gamma = 0.6$ in yellow, $\gamma = 0.8$ in green, $\gamma = 0.9$ in cyan, and $\gamma = 0.99$ in blue). Voxels correlated with two or more regressors are shown by a mosaic of colors. **(A)** Significant correlation with reward prediction $V(t)$ was observed in the MFC, DLPFC, dorsal striatum, insula, and hippocampus. Note the anterior-ventral to posterior-dorsal gradient with the increase in $\gamma$ in the MFC. **(B)** Significant correlation with reward prediction error $\delta(t)$ on $\gamma = 0$ was observed in the ventral striatum.

## 4   Discussion

In the MFC, anterior and ventral part was involved in reward prediction $V(t)$ on shorter time scales ($0 \leq \gamma \leq 0.6$), whereas posterior and dorsal part was involved in reward prediction $V(t)$ on longer time scales ($0.6 \leq \gamma \leq 0.99$). The ventral striatum involved in reward prediction error $\delta(t)$ on shortest time scale ($\gamma = 0$), while the dorsolateral striatum correlated with reward prediction $V(t)$ on longer time scales ($0.9 \leq \gamma \leq 0.99$). These results are consistent with the topographic organization of fronto-striatal connection; the rostral part of the MFC project to the ventral striatum, whereas the dorsal and posterior part of the cingulate cortex project to the dorsolateral striatum [21].

In the MFC and the striatum, no significant difference in activity was observed in block-design analysis while we did find graded maps of activities with different values of $\gamma$. A possible reason is that different parts of the MFC and the striatum are concurrently involved with reward prediction on different time scales, regardless of the task context. Activities of the DLPFC and lOFC, which show significant differences in block-design analysis (Fig. 3), may be regulated according to the necessity for the task;

From these results, we propose the following mechanism of reward prediction on different time scales. The parallel cortico-basal ganglia loops are responsible for reward prediction on various time scales. The 'limbic loop' via the ventral striatum specializes in immediate reward prediction, whereas the 'cognitive and motor loop' via the dorsal striatum specialises in future reward prediction. Each loop learns to predict rewards on its specific time scale. To perform an optimal action under a given time scale, the output of the loop with an appropriate time scale is used for actual action selection.

Previous studies in brain damages and serotonergic functions suggest that the MFC and the dorsal raphe, which are reciprocally connected [22, 23], play an important role in future reward prediction. The cortico-cortico projections from the MFC, or the serotonergic projections from the dorsal raphe to the cortex and the striatum may be involved in the modulation of these parallel loops.

In present study, using a novel regression analysis based on subjects' performance data and reinforcement learning model, we revealed the maps of time scales in reward prediction, which could not be found by conventional block-design analysis. Future studies using this method under pharmacological manipulation of the serotonergic system would clarify the role of serotonin in regulating the time scale of reward prediction.

## Acknowledgments

We thank Nicolas Schweighofer, Kazuyuki Samejima, Masahiko Haruno, Hiroshi Imamizu, Satomi Higuchi, Toshinori Yoshioka, and Mitsuo Kawato for helpful discussions and technical advice.

## References

[1] Rogers, R.D., et al. (1999) Dissociable deficits in the decision-making cognition of chronic amphetamine abusers, opiate abusers, patients with focal damage to prefrontal cortex, and tryptophan-depleted normal volunteers: evidence for monoaminergic mechanisms. Neuropsychopharmacology 20(4):322-339.

[2] Evenden, J.L. & Ryan, C.N. (1996) The pharmacology of impulsive behaviour in rats: the effects of drugs on response choice with varying delays of reinforcement. Psychopharmacology (Berl) 128(2):161-170.

[3] Mobini, S., et al. (2000) Effects of central 5-hydroxytryptamine depletion on sensitivity to delayed and probabilistic reinforcement. Psychopharmacology (Berl) 152(4):390-397.

[4] Bechara, A., et al. (1994) Insensitivity to future consequences following damage to human prefrontal cortex. Cognition 50(1-3):7-15.

[5] Bechara, A., Tranel, D. & Damasio, H. (2000) Characterization of the decision-making deficit of patients with ventromedial prefrontal cortex lesions. Brain 123:2189-2202.

[6] Mobini, S., et al. (2002) Effects of lesions of the orbitofrontal cortex on sensitivity to delayed and probabilistic reinforcement. Psychopharmacology (Berl) 160(3):290-298.

[7] Doya, K. (2002) Metalearning and neuromodulation. Neural Netw 15(4-6):495-506.

[8] Sutton, R.S., Barto, A. G. (1998) Reinforcement learning. Cambridge, MA: MIT press.

[9] Houk, J.C., Adams, J.L. & Barto, A.G., A model of how the basal ganglia generate and use neural signals that predict reinforcement, in Models of information processing in the basal ganglia, J.C. Houk, J.L. Davis, and D.G. Beiser, Editors. 1995, MIT Press: Cambridge, Mass. p. 249-270.

[10] Schultz, W., Dayan, P. & Montague, P.R. (1997) A neural substrate of prediction and reward. Science 275(5306):1593-1599.

[11] Doya, K. (2000) Complementary roles of basal ganglia and cerebellum in learning and motor control. Curr Opin Neurobiol 10(6):732-739.

[12] Berns, G.S., et al. (2001) Predictability modulates human brain response to reward. J Neurosci 21(8):2793-2798.

[13] O'Doherty, J.P., et al. (2003) Temporal difference models and reward-related learning in the human brain. Neuron 38(2):329-337.

[14] Koepp, M.J., et al. (1998) Evidence for striatal dopamine release during a video game. Nature 393(6682):266-268.

[15] Rogers, R.D., et al. (1999) Choosing between small, likely rewards and large, unlikely rewards activates inferior and orbital prefrontal cortex. J Neurosci 19(20):9029-9038.

[16] Elliott, R., Friston, K.J. & Dolan, R.J. (2000) Dissociable neural responses in human reward systems. J Neurosci 20(16):6159-6165.

[17] Breiter, H.C., et al. (2001) Functional imaging of neural responses to expectancy and experience of monetary gains and losses. Neuron 30(2):619-639.

[18] Knutson, B., et al. (2001) Anticipation of increasing monetary reward selectively recruits nucleus accumbens. J Neurosci 21(16):RC159.

[19] O'Doherty, J.P., et al. (2002) Neural responses during anticipation of a primary taste reward. Neuron 33(5):815-826.

[20] Pagnoni, G., et al. (2002) Activity in human ventral striatum locked to errors of reward prediction. Nat Neurosci 5(2):97-98.

[21] Haber, S.N., et al. (1995) The orbital and medial prefrontal circuit through the primate basal ganglia. J Neurosci 15(7 Pt 1):4851-4867.

[22] Celada, P., et al. (2001) Control of dorsal raphe serotonergic neurons by the medial prefrontal cortex: Involvement of serotonin-1A, GABA(A), and glutamate receptors. J Neurosci 21(24):9917-9929.

[23] Martin-Ruiz, R., et al. (2001) Control of serotonergic function in medial pre-frontal cortex by serotonin-2A receptors through a glutamate-dependent mechanism. J Neurosci 21(24):9856-9866.
